# Inductive reasoning about chimeric creatures

**Charles Kemp**
Department of Psychology
Carnegie Mellon University
ckemp@cmu.edu

## Abstract

Given one feature of a novel animal, humans readily make inferences about other features of the animal. For example, winged creatures often fly, and creatures that eat fish often live in the water. We explore the knowledge that supports these inferences and compare two approaches. The first approach proposes that humans rely on abstract representations of dependency relationships between features, and is formalized here as a graphical model. The second approach proposes that humans rely on specific knowledge of previously encountered animals, and is formalized here as a family of exemplar models. We evaluate these models using a task where participants reason about chimeras, or animals with pairs of features that have not previously been observed to co-occur. The results support the hypothesis that humans rely on explicit representations of relationships between features.

Suppose that an eighteenth-century naturalist learns about a new kind of animal that has fur and a duck's bill. Even though the naturalist has never encountered an animal with this pair of features, he should be able to make predictions about other features of the animal—for example, the animal could well live in water but probably does not have feathers. Although the platypus exists in reality, from a eighteenth-century perspective it qualifies as a *chimera*, or an animal that combines two or more features that have not previously been observed to co-occur. Here we describe a probabilistic account of inductive reasoning and use it to account for human inferences about chimeras.

The inductive problems we consider are special cases of the more general problem in Figure 1a where a reasoner is given a partially observed matrix of animals by features then asked to infer the values of the missing entries. This general problem has been previously studied and is addressed by computational models of property induction, categorization, and generalization [1–7]. A challenge faced by all of these models is to capture the background knowledge that guides inductive inferences. Some accounts rely on similarity relationships between animals [6, 8], others rely on causal relationships between features [9, 10], and others incorporate relationships between animals and relationships between features [11]. We will evaluate graphical models that capture both kinds of relationships (Figure 1a), but will focus in particular on relationships between features.

Psychologists have previously suggested that humans rely on explicit mental representations of relationships between features [12–16]. Often these representations are described as theories—for example, theories that specify a causal relationship between having wings and flying, or living in the sea and eating fish. Relationships between features may take several forms: for example, one feature may cause, enable, prevent, be inconsistent with, or be a special case of another feature. For simplicity, we will treat all of these relationships as instances of dependency relationships between features, and will capture them using an undirected graphical model.

Previous studies have used graphical models to account for human inferences about features but typically these studies consider toy problems involving a handful of novel features such as "has gene X14" or "has enzyme Y132" [9, 11]. Participants might be told, for example, that gene X14 leads to the production of enzyme Y132, then asked to use this information when reasoning about novel animals. Here we explore whether a graphical model approach can account for inferences

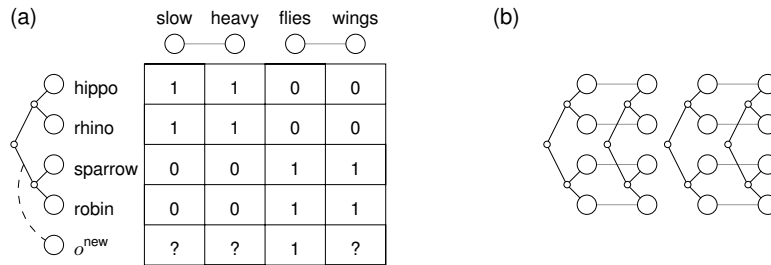

| (a) | | slow | heavy | flies | wings | (b) |
|---|---|---|---|---|---|---|
| hippo | | 1 | 1 | 0 | 0 | |
| rhino | | 1 | 1 | 0 | 0 | |
| sparrow | | 0 | 0 | 1 | 1 | |
| robin | | 0 | 0 | 1 | 1 | |
| $o^{\text{new}}$ | | ? | ? | 1 | ? | |

Figure 1: Inductive reasoning about animals and features. (a) Inferences about the features of a new animal $o^{\text{new}}$ that flies may draw on similarity relationships between animals (the new animal is similar to sparrows and robins but not hippos and rhinos), and on dependency relationships between features (flying and having wings are linked). (b) A graph product produced by combining the two graph structures in (a).

about familiar features. Working with familiar features raises a methodological challenge since participants have a substantial amount of knowledge about these features and can reason about them in multiple ways. Suppose, for example, that you learn that a novel animal can fly (Figure 1a). To conclude that the animal probably has wings, you might consult a mental representation similar to the graph at the top of Figure 1a that specifies a dependency relationship between flying and having wings. On the other hand, you might reach the same conclusion by thinking about flying creatures that you have previously encountered (e.g. sparrows and robins) and noticing that these creatures have wings. Since the same conclusion can be reached in two different ways, judgments about arguments of this kind provide little evidence about the mental representations involved.

The challenge of working with familiar features directly motivates our focus on chimeras. Inferences about chimeras draw on rich background knowledge but require the reasoner to go beyond past experience in a fundamental way. For example, if you learn that an animal flies and has no legs, you cannot make predictions about the animal by thinking of flying, no-legged creatures that you have previously encountered. You may, however, still be able to infer that the novel animal has wings if you understand the relationship between flying and having wings. We propose that graphical models over features can help to explain how humans make inferences of this kind, and evaluate our approach by comparing it to a family of exemplar models. The next section introduces these models, and we then describe two experiments designed to distinguish between the models.

## 1 Reasoning about objects and features

Our models make use of a binary matrix $D$ where the rows $\{o^1, \ldots, o^{129}\}$ correspond to objects, and the columns $\{f^1, \ldots, f^{56}\}$ correspond to features. A subset of the objects is shown in Figure 2a, and the full set of features is shown in Figure 2b and its caption. Matrix $D$ was extracted from the Leuven natural concept database [17], which includes 129 animals and 757 features in total. We chose a subset of these features that includes a mix of perceptual and behavioral features, and that includes many pairs of features that depend on each other. For example, animals that "live in water" typically "can swim," and animals that have "no legs" cannot "jump far."

Matrix $D$ can be used to formulate problems where a reasoner observes one or two features of a new object (i.e. animal $o^{130}$) and must make inferences about the remaining features of the animal. The next two sections describe graphical models that can be used to address this problem. The first graphical model $\mathcal{O}$ captures relationships between objects, and the second model $\mathcal{F}$ captures relationships between features. We then discuss how these models can be combined, and introduce a family of exemplar-style models that will be compared with our graphical models.

**A graphical model over objects**

Many accounts of inductive reasoning focus on similarity relationships between objects [6, 8]. Here we describe a tree-structured graphical model $\mathcal{O}$ that captures these relationships. The tree was constructed from matrix $D$ using average linkage clustering and the Jaccard similarity measure, and part of the resulting structure is shown in Figure 2a. The subtree in Figure 2a includes clusters

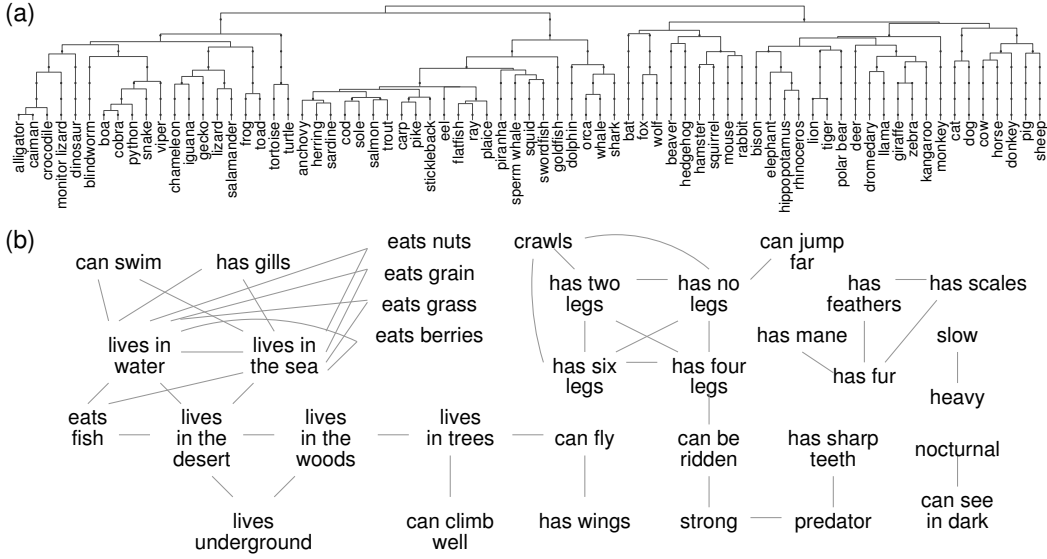

Figure 2: Graph structures used to define graphical models $\mathcal{O}$ and $\mathcal{F}$. (a) A tree that captures similarity relationships between animals. The full tree includes 129 animals, and only part of the tree is shown here. The grey points along the branches indicate locations where a novel animal $o^{130}$ could be attached to the tree. (b) A network capturing pairwise dependency relationships between features. The edges capture both positive and negative dependencies. All edges in the network are shown, and the network also includes 20 isolated nodes for the following features: is black, is blue, is green, is grey, is pink, is red, is white, is yellow, is a pet, has a beak, stings, stinks, has a long neck, has feelers, sucks blood, lays eggs, makes a web, has a hump, has a trunk, and is cold-blooded.

corresponding to amphibians and reptiles, aquatic creatures, and land mammals, and the subtree omitted for space includes clusters for insects and birds.

We assume that the features in matrix $D$ (i.e. the columns) are generated independently over $\mathcal{O}$:

$$P(D|\mathcal{O}, \pi, \lambda) = \prod_i P(f^i|\mathcal{O}, \pi^i, \lambda^i).$$

The distribution $P(f^i|\mathcal{O}, \pi^i, \lambda^i)$ is based on the intuition that nearby nodes in $\mathcal{O}$ tend to have the same value of $f^i$. Previous researchers [8, 18] have used a directed graphical model where the distribution at the root node is based on the baserate $\pi^i$, and any other node $v$ with parent $u$ has the following conditional probability distribution:

$$P(v = 1|u) = \begin{cases} \pi^i + (1 - \pi^i)e^{-\lambda^i l}, & \text{if } u = 1 \\ \pi^i - \pi^i e^{-\lambda^i l}, & \text{if } u = 0 \end{cases} \tag{1}$$

where $l$ is the length of the branch joining node $u$ to node $v$. The variability parameter $\lambda^i$ captures the extent to which feature $f^i$ is expected to vary over the tree. Note, for example, that any node $v$ must take the same value as its parent $u$ when $\lambda = 0$. To avoid free parameters, the feature baserates $\pi^i$ and variability parameters $\lambda^i$ are set to their maximum likelihood values given the observed values of the features $\{f^i\}$ in the data matrix $D$. The conditional distributions in Equation 1 induce a joint distribution over all of the nodes in graph $\mathcal{O}$, and the distribution $P(f^i|\mathcal{O}, \pi^i, \lambda^i)$ is computed by marginalizing out the values of the internal nodes. Although we described $\mathcal{O}$ as a directed graphical model, the model can be converted into an equivalent undirected model with a potential for each edge in the tree and a potential for the root node. Here we use the undirected version of the model, which is a natural counterpart to the undirected model $\mathcal{F}$ described in the next section.

The full version of structure $\mathcal{O}$ in Figure 2a includes 129 familiar animals, and our task requires inferences about a novel animal $o^{130}$ that must be slotted into the structure. Let $D'$ be an expanded version of $D$ that includes a row for $o^{130}$, and let $\mathcal{O}'$ be an expanded version of $\mathcal{O}$ that includes a node for $o^{130}$. The edges in Figure 2a are marked with evenly spaced gray points, and we use a

uniform prior $P(\mathcal{O}')$ over all trees that can be created by attaching $o^{130}$ to one of these points. Some of these trees have identical topologies, since some edges in Figure 2a have multiple gray points. Predictions about $o^{130}$ can be computed using:

$$P(D'|D) = \sum_{\mathcal{O}'} P(D'|\mathcal{O}', D)P(\mathcal{O}'|D) \propto \sum_{\mathcal{O}'} P(D'|\mathcal{O}', D)P(D|\mathcal{O}')P(\mathcal{O}'). \qquad (2)$$

Equation 2 captures the basic intuition that the distribution of features for $o^{130}$ is expected to be consistent with the distribution observed for previous animals. For example, if $o^{130}$ is known to fly then the trees with high posterior probability $P(\mathcal{O}'|D)$ will be those where $o^{130}$ is near other flying creatures (Figure 1a), and since these creatures have wings Equation 2 predicts that $o^{130}$ probably also has wings. As this example suggests, model $\mathcal{O}$ captures dependency relationships between features implicitly, and therefore stands in contrast to models like $\mathcal{F}$ that rely on explicit representations of relationships between features.

**A graphical model over features**

Model $\mathcal{F}$ is an undirected graphical model defined over features. The graph shown in Figure 2b was created by identifying pairs where one feature depends directly on another. The author and a research assistant both independently identified candidate sets of pairwise dependencies, and Figure 2b was created by merging these sets and reaching agreement about how to handle any discrepancies.

As previous researchers have suggested [13, 15], feature dependencies can capture several kinds of relationships. For example, wings *enable* flying, living in the sea *leads to* eating fish, and having no legs *rules out* jumping far. We work with an undirected graph because some pairs of features depend on each other but there is no clear direction of causal influence. For example, there is clearly a dependency relationship between being nocturnal and seeing in the dark, but no obvious sense in which one of these features causes the other.

We assume that the rows of the object-feature matrix $D$ are generated independently from an undirected graphical model $\mathcal{F}$ defined over the feature structure in Figure 2b:

$$P(D|\mathcal{F}) = \prod_i P(o^i|\mathcal{F}).$$

Model $\mathcal{F}$ includes potential functions for each node and for each edge in the graph. These potentials were learned from matrix $D$ using the UGM toolbox for undirected graphical models [19]. The learned potentials capture both positive and negative relationships: for example, animals that live in the sea tend to eat fish, and tend not to eat berries. Some pairs of feature values never occur together in matrix $D$ (there are no creatures that fly but do not have wings). We therefore chose to compute maximum *a posteriori* values of the potential functions rather than maximum likelihood values, and used a diffuse Gaussian prior with a variance of 100 on the entries in each potential.

After learning the potentials for model $\mathcal{F}$, we can make predictions about a new object $o^{130}$ using the distribution $P(o^{130}|\mathcal{F})$. For example, if $o^{130}$ is known to fly (Figure 1a), model $\mathcal{F}$ predicts that $o^{130}$ probably has wings because the learned potentials capture a positive dependency between flying and having wings.

**Combining object and feature relationships**

There are two simple ways to combine models $\mathcal{O}$ and $\mathcal{F}$ in order to develop an approach that incorporates both relationships between features and relationships between objects. The *output combination* model computes the predictions of both models in isolation, then combines these predictions using a weighted sum. The resulting model is similar to a mixture-of-experts model, and to avoid free parameters we use a mixing weight of 0.5. The *structure combination* model combines the graph structures used by the two models and relies on a set of potentials defined over the resulting graph product. An example of a graph product is shown in Figure 1b, and the potential functions for this graph are inherited from the component models in the natural way. Kemp et al. [11] use a similar approach to combine a functional causal model with an object model $\mathcal{O}$, but note that our structure combination model uses an undirected model $F$ rather than a functional causal model over features.

Both combination models capture the intuition that inductive inferences rely on relationships between features and relationships between objects. The output combination model has the virtue of

simplicity, and the structure combination model is appealing because it relies on a single integrated representation that captures both relationships between features and relationships between objects. To preview our results, our data suggest that the combination models perform better overall than either $\mathcal{O}$ or $\mathcal{F}$ in isolation, and that both combination models perform about equally well.

**Exemplar models**

We will compare the family of graphical models already described with a family of exemplar models. The key difference between these model families is that the exemplar models do not rely on explicit representations of relationships between objects and relationships between features. Comparing the model families can therefore help to establish whether human inferences rely on representations of this sort.

Consider first a problem where a reasoner must predict whether object $o^{130}$ has feature $k$ after observing that it has feature $i$. An exemplar model addresses the problem by retrieving all previously-observed objects with feature $i$ and computing the proportion that have feature $k$:

$$P(o_k = 1 | o_i = 1) = \frac{|f^k \& f^i|}{|f^i|} \tag{3}$$

where $|f^k|$ is the number of objects in matrix $D$ that have feature $k$, and $|f^k \& f^i|$ is the number that have both feature $k$ and feature $i$. Note that we have streamlined our notation by using $o_k$ instead of $o_k^{130}$ to refer to the $k$th feature value for object $o^{130}$.

Suppose now that the reasoner observes that object $o^{130}$ has features $i$ and $j$. The natural generalization of Equation 3 is:

$$P(o_k = 1 | o_i = 1, o_j = 1) = \frac{|f^k \& f^i \& f^j|}{|f^i \& f^j|} \tag{4}$$

Because we focus on chimeras, $|f^i \& f^j| = 0$ and Equation 4 is not well defined. We therefore evaluate an exemplar model that computes predictions for the two observed features separately then computes the weighted sum of these predictions:

$$P(o_k = 1 | o_i = 1, o_j = 1) = w^i \frac{|f^k \& f^i|}{|f^i|} + w^j \frac{|f^k \& f^j|}{|f^j|}. \tag{5}$$

where the weights $w^i$ and $w^j$ must sum to one. We consider four ways in which the weights could be set. The first strategy sets $w^i = w^j = 0.5$. The second strategy sets $w^i \propto |f^i|$, and is consistent with an approach where the reasoner retrieves all exemplars in $D$ that are most similar to the novel animal and reports the proportion of these exemplars that have feature $k$. The third strategy sets $w^i \propto \frac{1}{|f^i|}$, and captures the idea that features should be weighted by their distinctiveness [20]. The final strategy sets weights according to the *coherence* of each feature [21]. A feature is coherent if objects with that feature tend to resemble each other overall, and we define the coherence of feature $i$ as the expected Jaccard similarity between two randomly chosen objects from matrix $D$ that both have feature $i$. Note that the final three strategies are all consistent with previous proposals from the psychological literature, and each one might be expected to perform well.

Because exemplar models and prototype models are often compared, it is natural to consider a prototype model [22] as an additional baseline. A standard prototype model would partition the 129 animals into categories and would use summary statistics for these categories to make predictions about the novel animal $o^{130}$. We will not evaluate this model because it corresponds to a coarser version of model $\mathcal{O}$, which organizes the animals into a hierarchy of categories. The key characteristic shared by both models is that they explicitly capture relationships between objects but not features.

## 2 Experiment 1: Chimeras

Our first experiment explores how people make inferences about chimeras, or novel animals with features that have not previously been observed to co-occur. Inferences about chimeras raise challenges for exemplar models, and therefore help to establish whether humans rely on explicit representations of relationships between features. Each argument can be represented as $f^i, f^j \to f^k$

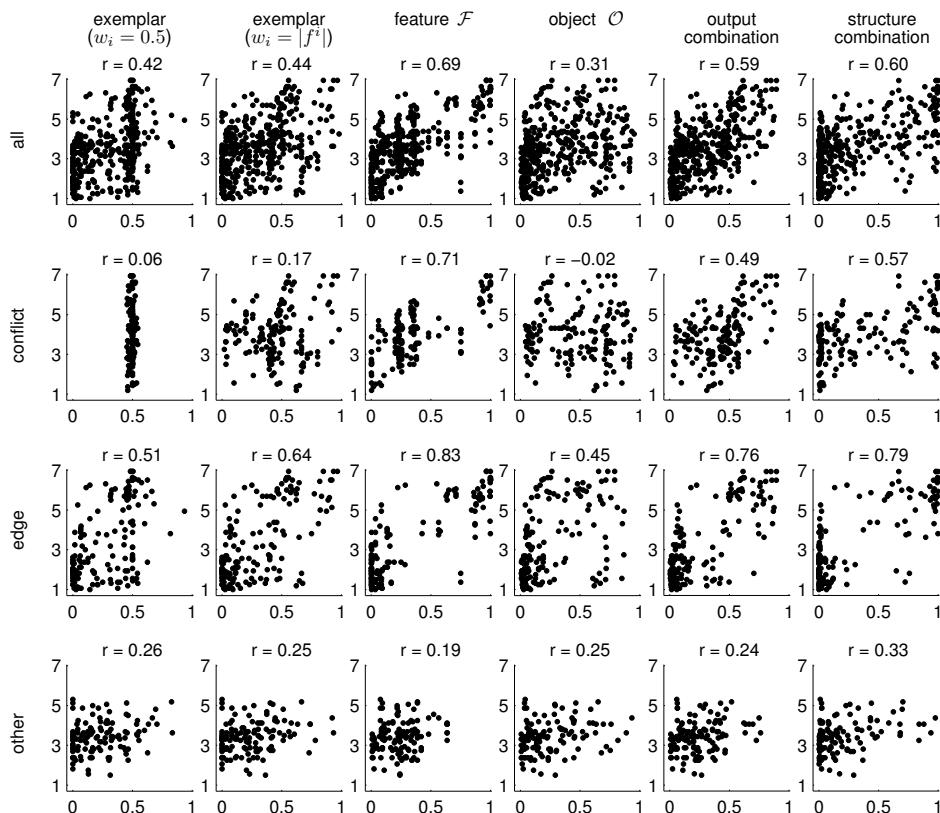

Figure 3: Argument ratings for Experiment 1 plotted against the predictions of six models. The y-axis in each panel shows human ratings on a seven point scale, and the x-axis shows probabilities according to one of the models. Correlation coefficients are shown for each plot.

where $f^i$ and $f^k$ are the premises (e.g. "has no legs" and "can fly") and $f^k$ is the conclusion (e.g. "has wings"). We are especially interested in conflict cases where the premises $f^i$ and $f^j$ lead to opposite conclusions when taken individually: for example, most animals with no legs do not have wings, but most animals that fly do have wings. Our models that incorporate feature structure $\mathcal{F}$ can resolve this conflict since $\mathcal{F}$ includes a dependency between "wings" and "can fly" but not between "wings" and "has no legs." Our models that do not include $\mathcal{F}$ cannot resolve the conflict and predict that humans will be uncertain about whether the novel animal has wings.

**Materials.** The object-feature matrix $D$ includes 447 feature pairs $\{f^i, f^j\}$ such that none of the 129 animals has both $f^i$ and $f^j$. We selected 40 pairs (see the supporting material) and created 400 arguments in total by choosing 10 conclusion features for each pair. The arguments can be assigned to three categories. *Conflict* cases are arguments $f^i, f^j \rightarrow f^k$ such that the single-premise arguments $f^i \rightarrow f^k$ and $f^j \rightarrow f^k$ lead to incompatible predictions. For our purposes, two single-premise arguments with the same conclusion are deemed incompatible if one leads to a probability greater than 0.9 according to Equation 3, and the other leads to a probability less than 0.1. *Edge* cases are arguments $f^i, f^j \rightarrow f^k$ such that the feature network in Figure 2b includes an edge between $f^k$ and either $f^i$ or $f^j$. Note that some arguments are both conflict cases and edge cases. All arguments that do not fall into either one of these categories will be referred to as *other* cases.

The 400 arguments for the experiment include 154 conflict cases, 153 edge cases, and 120 other cases. 34 arguments are both conflict cases and edge cases. We chose these arguments based on three criteria. First, we avoided premise pairs that did not co-occur in matrix $D$ but that co-occur in familiar animals that do not belong to $D$. For example, "is pink" and "has wings" do not co-occur in $D$ but "flamingo" is a familiar animal that has both features. Second, we avoided premise pairs that specified two different numbers of legs—for example, {"has four legs," "has six legs"}. Finally, we aimed to include roughly equal numbers of conflict cases, edge cases, and other cases.

**Method.** 16 undergraduates participated for course credit. The experiment was carried out using a custom-built computer interface, and one argument was presented on screen at a time. Participants

rated the probability of the conclusion on seven point scale where the endpoints were labeled "very unlikely" and "very likely." The ten arguments for each pair of premises were presented in a block, but the order of these blocks and the order of the arguments within these blocks were randomized across participants.

**Results.** Figure 3 shows average human judgments plotted against the predictions of six models. The plots in the first row include all 400 arguments in the experiment, and the remaining rows show results for conflict cases, edge cases, and other cases. The previous section described four exemplar models, and the two shown in Figure 3 are the best performers overall. Even though the graphical models include more numerical parameters than the exemplar models, recall that these parameters are learned from matrix $D$ rather than fit to the experimental data. Matrix $D$ also serves as the basis for the exemplar models, which means that all of the models can be compared on equal terms.

The first row of Figure 3 suggests that the three models which include feature structure $\mathcal{F}$ perform better than the alternatives. The output combination model is the worst of the three models that incorporate $\mathcal{F}$, and the correlation achieved by this model is significantly greater than the correlation achieved by the best exemplar model ($p < 0.001$, using the Fisher transformation to convert correlation coefficients to z scores). Our data therefore suggest that explicit representations of relationships between features are needed to account for inductive inferences about chimeras. The model that includes the feature structure $\mathcal{F}$ alone performs better than the two models that combine $\mathcal{F}$ with the object structure $\mathcal{O}$, which may not be surprising since Experiment 1 focuses specifically on novel animals that do not slot naturally into structure $\mathcal{O}$.

Rows two through four suggest that the conflict arguments in particular raise challenges for the models which do not include feature structure $\mathcal{F}$. Since these conflict cases are arguments $f^i, f^j \rightarrow f^k$ where $f^i \rightarrow f^k$ has strength greater than 0.9 and $f^j \rightarrow f^k$ has strength less than 0.1, the first exemplar model averages these strengths and assigns an overall strength of around 0.5 to each argument. The second exemplar model is better able to differentiate between the conflict arguments, but still performs substantially worse than the three models that include structure $\mathcal{F}$. The exemplar models perform better on the edge arguments, but are outperformed by the models that include $\mathcal{F}$. Finally, all models achieve roughly the same level of performance on the other arguments.

Although the feature model $\mathcal{F}$ performs best overall, the predictions of this model still leave room for improvement. The two most obvious outliers in the third plot in the top row represent the arguments {is blue, lives in desert → lives in woods} and {is pink, lives in desert → lives in woods}. Our participants sensibly infer that any animal which lives in the desert cannot simultaneously live in the woods. In contrast, the Leuven database indicates that eight of the twelve animals that live in the desert also live in the woods, and the edge in Figure 2b between "lives in the desert" and "lives in the woods" therefore represents a positive dependency relationship according to model $\mathcal{F}$. This discrepancy between model and participants reflects the fact that participants made inferences about individual animals but the Leuven database is based on features of animal categories. Note, for example, that any individual animal is unlikely to live in the desert and the woods, but that some animal categories (including snakes, salamanders, and lizards) are found in both environments.

## 3   Experiment 2: Single-premise arguments

Our results so far suggest that inferences about chimeras rely on explicit representations of relationships between features but provide no evidence that relationships between objects are important. It would be a mistake, however, to conclude that relationships between objects play no role in inductive reasoning. Previous studies have used object structures like the example in Figure 2a to account for inferences about novel features [11]—for example, given that alligators have enzyme Y132 in their blood, it seems likely that crocodiles also have this enzyme. Inferences about novel objects can also draw on relationships between objects rather than relationships between features. For example, given that a novel animal has a beak you will probably predict that it has feathers, not because there is any direct dependency between these two features, but because the beaked animals that you know tend to have feathers. Our second experiment explores inferences of this kind.

**Materials and Method.** 32 undergraduates participated for course credit. The task was identical to Experiment 1 with the following exceptions. Each two-premise argument $f^i, f^j \rightarrow f^k$ from Experiment 1 was converted into two one-premise arguments $f^i \rightarrow f^k$ and $f^j \rightarrow f^k$, and these

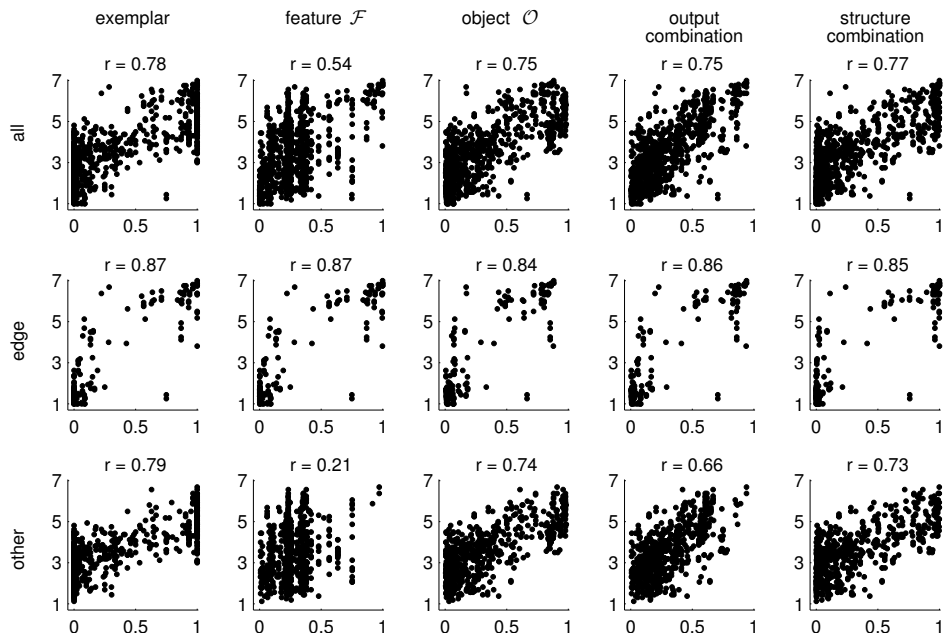

Figure 4: Argument ratings and model predictions for Experiment 2.

one-premise arguments were randomly assigned to two sets. 16 participants rated the 400 arguments in the first set, and the other 16 rated the 400 arguments in the second set.

**Results.** Figure 4 shows average human ratings for the 800 arguments plotted against the predictions of five models. Unlike Figure 3, Figure 4 includes a single exemplar model since there is no need to consider different feature weightings in this case. Unlike Experiment 1, the feature model $\mathcal{F}$ performs worse than the other alternatives ($p < 0.001$ in all cases). Not surprisingly, this model performs relatively well for edge cases $f^j \rightarrow f^k$ where $f^j$ and $f^k$ are linked in Figure 2b, but the final row shows that the model performs poorly across the remaining set of arguments.

Taken together, Experiments 1 and 2 suggest that relationships between objects and relationships between features are both needed to account for human inferences. Experiment 1 rules out an exemplar approach but models that combine graph structures over objects and features perform relatively well in both experiments. We considered two methods for combining these structures and both performed equally well. Combining the knowledge captured by these structures appears to be important, and future studies can explore in detail how humans achieve this combination.

# 4   Conclusion

This paper proposed that graphical models are useful for capturing knowledge about animals and their features and showed that a graphical model over features can account for human inferences about chimeras. A family of exemplar models and a graphical model defined over objects were unable to account for our data, which suggests that humans rely on mental representations that explicitly capture dependency relationships between features. Psychologists have previously used graphical models to capture relationships between features, but our work is the first to focus on chimeras and to explore models defined over a large set of familiar features.

Although a simple undirected model accounted relatively well for our data, this model is only a starting point. The model incorporates dependency relationships between features, but people know about many specific kinds of dependencies, including cases where one feature causes, enables, prevents, or is inconsistent with another. An undirected graph with only one class of edges cannot capture this knowledge in full, and richer representations will ultimately be needed in order to provide a more complete account of human reasoning.

**Acknowledgments**   I thank Madeleine Clute for assisting with this research. This work was supported in part by the Pittsburgh Life Sciences Greenhouse Opportunity Fund and by NSF grant CDI-0835797.

# References

[1] R. N. Shepard. Towards a universal law of generalization for psychological science. *Science*, 237:1317–1323, 1987.

[2] J. R. Anderson. The adaptive nature of human categorization. *Psychological Review*, 98(3):409–429, 1991.

[3] E. Heit. A Bayesian analysis of some forms of inductive reasoning. In M. Oaksford and N. Chater, editors, *Rational models of cognition*, pages 248–274. Oxford University Press, Oxford, 1998.

[4] J. B. Tenenbaum and T. L. Griffiths. Generalization, similarity, and Bayesian inference. *Behavioral and Brain Sciences*, 24:629–641, 2001.

[5] C. Kemp and J. B. Tenenbaum. Structured statistical models of inductive reasoning. *Psychological Review*, 116(1):20–58, 2009.

[6] D. N. Osherson, E. E. Smith, O. Wilkie, A. Lopez, and E. Shafir. Category-based induction. *Psychological Review*, 97(2):185–200, 1990.

[7] D. J. Navarro. Learning the context of a category. In J. Lafferty, C. K. I. Williams, J. Shawe-Taylor, R.S. Zemel, and A. Culotta, editors, *Advances in Neural Information Processing Systems 23*, pages 1795–1803. 2010.

[8] C. Kemp, T. L. Griffiths, S. Stromsten, and J. B. Tenenbaum. Semi-supervised learning with trees. In *Advances in Neural Information Processing Systems 16*, pages 257–264. MIT Press, Cambridge, MA, 2004.

[9] B. Rehder. A causal-model theory of conceptual representation and categorization. *Journal of Experimental Psychology: Learning, Memory, and Cognition*, 29:1141–1159, 2003.

[10] B. Rehder and R. Burnett. Feature inference and the causal structure of categories. *Cognitive Psychology*, 50:264–314, 2005.

[11] C. Kemp, P. Shafto, and J. B. Tenenbaum. An integrated account of generalization across objects and features. *Cognitive Psychology*, in press.

[12] S. E. Barrett, H. Abdi, G. L. Murphy, and J. McCarthy Gallagher. Theory-based correlations and their role in children's concepts. *Child Development*, 64:1595–1616, 1993.

[13] S. A. Sloman, B. C. Love, and W. Ahn. Feature centrality and conceptual coherence. *Cognitive Science*, 22(2):189–228, 1998.

[14] D. Yarlett and M. Ramscar. A quantitative model of counterfactual reasoning. In T. G. Dietterich, S. Becker, and Z. Ghahramani, editors, *Advances in Neural Information Processing Systems 14*, pages 123–130. MIT Press, Cambridge, MA, 2002.

[15] W. Ahn, J. K. Marsh, C. C. Luhmann, and K. Lee. Effect of theory-based feature correlations on typicality judgments. *Memory and Cognition*, 30(1):107–118, 2002.

[16] D. C. Meehan C. McNorgan, R. A. Kotack and K. McRae. Feature-feature causal relations and statistical co-occurrences in object concepts. *Memory and Cognition*, 35(3):418–431, 2007.

[17] S. De Deyne, S. Verheyen, E. Ameel, W. Vanpaemel, M. J. Dry, W. Voorspoels, and G. Storms. Exemplar by feature applicability matrices and other Dutch normative data for semantic concepts. *Behavior Research Methods*, 40(4):1030–1048, 2008.

[18] J. P. Huelsenbeck and F. Ronquist. MRBAYES: Bayesian inference of phylogenetic trees. *Bioinformatics*, 17(8):754–755, 2001.

[19] M. Schmidt. UGM: A Matlab toolbox for probabilistic undirected graphical models. 2007. Available at `http://people.cs.ubc.ca/~schmidtm/Software/UGM.html`.

[20] L. J. Nelson and D. T. Miller. The distinctiveness effect in social categorization: you are what makes you unusual. *Psychological Science*, 6:246–249, 1995.

[21] A. L. Patalano, S. Chin-Parker, and B. H. Ross. The importance of being coherent: category coherence, cross-classification and reasoning. *Journal of memory and language*, 54:407–424, 2006.

[22] S. K. Reed. Pattern recognition and categorization. *Cognitive Psychology*, 3:393–407, 1972.

